# Solvable Models of Artificial Neural Networks

Sumio Watanabe
Information and Communication R & D Center
Ricoh Co., Ltd.
3-2-3, Shin-Yokohama, Kohoku-ku, Yokohama, 222 Japan
sumio@ipe.rdc.ricoh.co.jp

## Abstract

Solvable models of nonlinear learning machines are proposed, and learning in artificial neural networks is studied based on the theory of ordinary differential equations. A learning algorithm is constructed, by which the optimal parameter can be found without any recursive procedure. The solvable models enable us to analyze the reason why experimental results by the error backpropagation often contradict the statistical learning theory.

## 1  INTRODUCTION

Recent studies have shown that learning in artificial neural networks can be understood as statistical parametric estimation using the maximum likelihood method [1], and that their generalization abilities can be estimated using the statistical asymptotic theory [2]. However, as is often reported, even when the number of parameters is too large, the error for the testing sample is not so large as the theory predicts. The reason for such inconsistency has not yet been clarified, because it is difficult for the artificial neural network to find the global optimal parameter.

On the other hand, in order to analyze the nonlinear phenomena, exactly solvable models have been playing a central role in mathematical physics, for example, the K-dV equation, the Toda lattice, and some statistical models that satisfy the Yang-

Baxter equation[3].

This paper proposes the first solvable models in the nonlinear learning problem. We consider simple three-layered neural networks, and show that the parameters from the inputs to the hidden units determine the function space that is characterized by a differential equation. This fact means that optimization of the parameters is equivalent to optimization of the differential equation. Based on this property, we construct a learning algorithm by which the optimal parameters can be found without any recursive procedure. Experimental result using the proposed algorithm shows that the maximum likelihood estimator is not always obtained by the error backpropagation, and that the conventional statistical learning theory leaves much to be improved.

## 2 The Basic Structure of Solvable Models

Let us consider a function $f_{c,w}(x)$ given by a simple neural network with 1 input unit, $H$ hidden units, and 1 output unit,

$$f_{c,w}(x) = \sum_{i=1}^{H} c_i \varphi_{w_i}(x), \tag{1}$$

where both $c = \{c_i\}$ and $w = \{w_i\}$ are parameters to be optimized, $\varphi_{w_i}(x)$ is the output of the $i$-th hidden unit.

We assume that $\{\varphi_i(x) = \varphi_{w_i}(x)\}$ is a set of independent functions in $C^H$-class. The following theorem is the start point of this paper.

**Theorem 1** *The $H$-th order differential equation whose fundamental system of solution is $\{\varphi_i(x)\}$ and whose $H$-th order coefficient is 1 is uniquely given by*

$$(D_w g)(x) \equiv (-1)^H \frac{W_{H+1}(g, \varphi_1, \varphi_2, ..., \varphi_H)}{W_H(\varphi_1, \varphi_2, ..., \varphi_H)} = 0, \tag{2}$$

*where $W_H$ is the $H$-th order Wronskian,*

$$W_H(\varphi_1, \varphi_2, ..., \varphi_H) = \det \begin{vmatrix} \varphi_1 & \varphi_2 & \cdots & \varphi_H \\ \varphi_1^{(1)} & \varphi_2^{(1)} & \cdots & \varphi_H^{(1)} \\ \varphi_1^{(2)} & \varphi_2^{(2)} & \cdots & \varphi_H^{(2)} \\ \cdots & \cdots & \cdots & \cdots \\ \varphi_1^{(H-1)} & \varphi_2^{(H-1)} & \cdots & \varphi_H^{(H-1)} \end{vmatrix}.$$

For proof, see [4]. From this theorem, we have the following corollary.

**Corollary 1** *Let $g(x)$ be a $C^H$-class function. Then the following conditions for $g(x)$ and $w = \{w_i\}$ are equivalent.*

*(1) There exists a set $c = \{c_i\}$ such that $g(x) = \sum_{i=1}^{H} c_i \varphi_{w_i}(x)$.*

*(2) $(D_w g)(x) = 0$.*

**Example 1**  Let us consider a case, $\varphi_{w_i}(x) = \exp(w_i x)$.

$$g(x) = \sum_{i=1}^{H} c_i \exp(w_i x)$$

is equivalent to $\{D^H + p_1 D^{H-1} + p_2 D^{H-2} + \cdots + p_H\}g(x) = 0$, where $D = (d/dx)$ and a set $\{p_i\}$ is determined from $\{w_i\}$ by the relation,

$$z^H + p_1 z^{H-1} + p_2 z^{H-2} + \cdots + p_H = \prod_{i=1}^{H}(z - w_i) \quad (\forall z \in \mathbf{C}).$$

**Example 2  (RBF)**   A function $g(x)$ is given by radial basis functions,

$$g(x) = \sum_{i=1}^{H} c_i \exp\{-(x - w_i)^2\},$$

if and only if $e^{-x^2}\{D^H + p_1 D^{H-1} + p_2 D^{H-2} + \cdots + p_H\}(e^{x^2} g(x)) = 0$, where a set $\{p_i\}$ is determined from $\{w_i\}$ by the relation,

$$z^H + p_1 z^{H-1} + p_2 z^{H-2} + \cdots + p_H = \prod_{i=1}^{H}(z - 2w_i) \quad (\forall z \in \mathbf{C}).$$

Figure 1 shows a learning algorithm for the solvable models. When a target function $g(x)$ is given, let us consider the following function approximation problem.

$$g(x) = \sum_{i=1}^{H} c_i \varphi_{w_i}(x) + \epsilon(x). \tag{3}$$

Learning in the neural network is optimizing both $\{c_i\}$ and $\{w_i\}$ such that $\epsilon(x)$ is minimized for some error function. From the definition of $D_w$, eq. (3) is equivalent to $(D_w g)(x) = (D_w \epsilon)(x)$, where the term $(D_w g)(x)$ is independent of $c_i$. Therefore, if we adopt $\|D_w \epsilon\|$ as the error function to be minimized, $\{w_i\}$ is optimized by minimizing $\|D_w g\|$, independently of $\{c_i\}$, where $\|f\|^2 = \int |f(x)|^2 dx$. After $\|D_w g\|$ is minimized, we have $(D_{w^*} g)(x) \approx 0$, where $w^*$ is the optimized parameter. From the corollary 1, there exists a set $\{c_i^*\}$ such that $g(x) \approx \sum c_i^* \varphi_{w_i^*}(x)$, where $\{c_i^*\}$ can be found using the ordinary least square method.

## 3  Solvable Models

For a general function $\varphi_w$, the differential operator $D_w$ does not always have such a simple form as the above examples. In this section, we consider a linear operator $L$ such that the differential equation of $L\varphi_w$ has a simple form.

**Definition**   A neural network $\sum c_i \varphi_{w_i}(x)$ is called *solvable* if there exist functions $a$, $b$, and a linear operator $L$ such that

$$(L\varphi_{w_i})(x) = \exp(a(w_i)x + b(w_i)).$$

The following theorem shows that the optimal parameter of the solvable models can be found using the same algorithm as Figure 1.

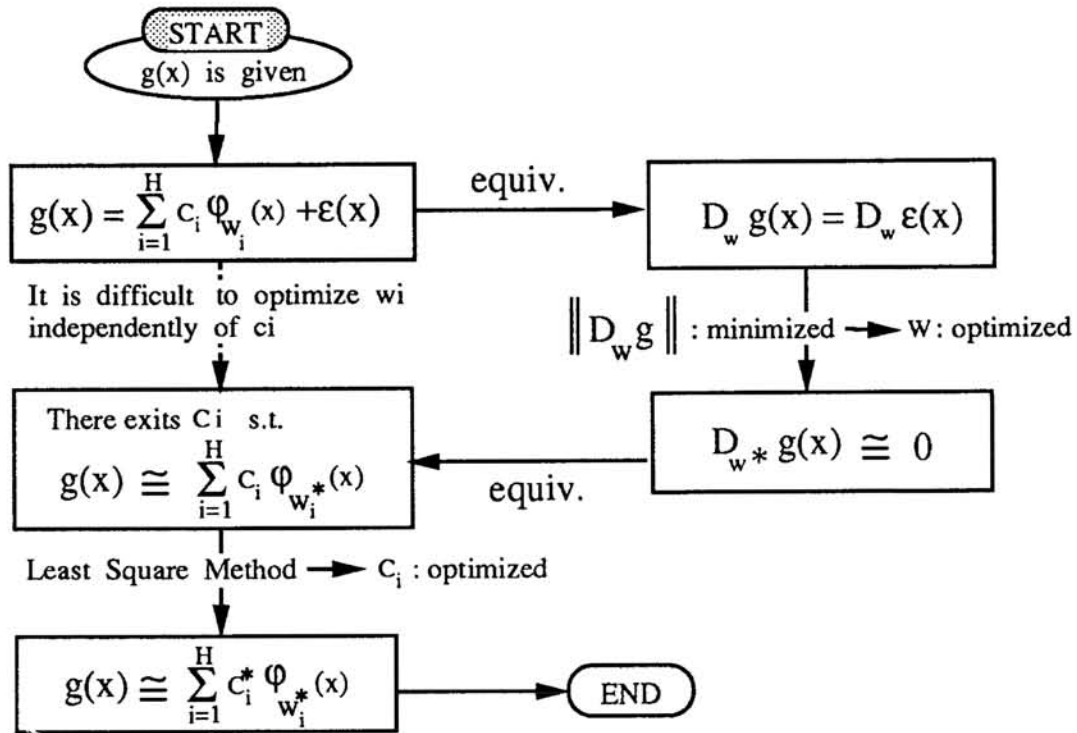

Figure 1: Structure of Solvable Models

**Theorem 2** *For a solvable model of a neural network, the following conditions are equivalent when $w_i \neq w_j$ $(i \neq j)$.*

*(1) There exist both $\{c_i\}$ and $\{w_i\}$ such that $g(x) = \sum_{i=1}^{H} c_i \varphi_{w_i}(x)$.*

*(2) There exists $\{p_i\}$ such that $\{D^H + p_1 D^{H-1} + p_2 D^{H-2} + \cdots + p_H\}(Lg)(x) = 0$.*

*(3) For arbitrary $Q > 0$, we define a sequence $\{y_n\}$ by $y_n = (Lg)(nQ)$. Then, there exists $\{q_i\}$ such that $y_n + q_1 y_{n-1} + q_2 y_{n-2} + \cdots + q_H y_{n-H} = 0$.*

Note that $\|D_w Lg\|^2$ is a quadratic form for $\{p_i\}$, which is easily minimized by the least square method. $\sum_n |y_n + q_1 y_{n-1} + \cdots + q_H y_{n-H}|^2$ is also a quadratic form for $\{q_i\}$.

**Theorem 3** *The sequences $\{w_i\}$, $\{p_i\}$, and $\{q_i\}$ in the theorem 2 have the following relations.*

$$z^H + p_1 z^{H-1} + p_2 z^{H-2} + \cdots + p_H = \prod_{i=1}^{H}(z - a(w_i)) \quad (\forall z \in \mathbf{C}),$$

$$z^H + q_1 z^{H-1} + q_2 z^{H-2} + \cdots + q_H = \prod_{i=1}^{H}(z - \exp(a(w_i)Q)) \quad (\forall z \in \mathbf{C}).$$

For proofs of the above theorems, see [5]. These theorems show that, if $\{p_i\}$ or

$\{q_i\}$ is optimized for a given function $g(x)$, then $\{a(w_i)\}$ can be found as a set of solutions of the algebraic equation.

Suppose that a target function $g(x)$ is given. Then, from the above theorems, the globally optimal parameter $w^* = \{w_i^*\}$ can be found by minimizing $\|D_w Lg\|$ independently of $\{c_i\}$. Moreover, if the function $a(w)$ is a one-to-one mapping, then there exists $w^*$ uniquely without permutation of $\{w_i^*\}$, if and only if the quadratic form $\|\{D^H + p_1 D^{H-1} + \cdots + p_H\}g\|^2$ is not degenerate[4]. (Remark that, if it is degenerate, we can use another neural network with the smaller number of hidden units.)

**Example 3**    A neural network without scaling

$$f_{b,c}(x) = \sum_{i=1}^{H} c_i \sigma(x + b_i), \tag{4}$$

is solvable when $(\mathcal{F}\sigma)(x) \neq 0$ (a.e.), where $\mathcal{F}$ denotes the Fourier transform. Define a linear operator $L$ by $(Lg)(x) = (\mathcal{F}g)(x)/(\mathcal{F}\sigma)(x)$, then, it follows that

$$(Lf_{b,c})(x) = \sum_{i=1}^{H} c_i \exp(-\sqrt{-1}\, b_i\, x). \tag{5}$$

By the Theorem 2, the optimal $\{b_i\}$ can be obtained by using the differential or the sequential equation.

**Example 4 (MLP)**    A three-layered perceptron

$$f_{b,c}(x) = \sum_{i=1}^{H} c_i \tan^{-1}\left(\frac{x + b_i}{a_i}\right), \tag{6}$$

is solvable. Define a linear operator $L$ by $(Lg)(x) = x \cdot (\mathcal{F}g)(x)$, then, it follows that

$$(Lf_{b,c})(x) = \sum_{i=1}^{H} c_i \exp(-(a_i + \sqrt{-1}\, b_i)x + \alpha(a_i, b_i)) \quad (x \geq 0). \tag{7}$$

where $\alpha(a_i, b_i)$ is some function of $a_i$ and $b_i$. Since the function $\tan^{-1}(x)$ is monotone increasing and bounded, we can expect that a neural network given by eq. (6) has the same ability in the function approximation problem as the ordinary three-layered perceptron using the sigmoid function, $\tanh(x)$.

**Example 5 (Finite Wavelet Decomposition)**    A finite wavelet decomposition

$$f_{b,c}(x) = \sum_{i=1}^{H} c_i \sigma\left(\frac{x + b_i}{a_i}\right), \tag{8}$$

is solvable when $\sigma(x) = (d/dx)^n(1/(1 + x^2))$ $(n \geq 1)$. Define a linear operator $L$ by $(Lg)(x) = x^{-n} \cdot (\mathcal{F}g)(x)$ then, it follows that

$$(Lf_{b,c})(x) = \sum_{i=1}^{H} c_i \exp(-(a_i + \sqrt{-1}\, b_i)x + \beta(a_i, b_i)) \quad (x \geq 0). \tag{9}$$

where $\beta(a_i, b_i)$ is some function of $a_i$ and $b_i$. Note that $\sigma(x)$ is an analyzing wavelet, and that this example shows a method how to optimize parameters for the finite wavelet decomposition.

## 4  Learning Algorithm

We construct a learning algorithm for solvable models, as shown in Figure 1.

<<Learning Algorithm>>
(0) A target function $g(x)$ is given.
(1) $\{y_m\}$ is calculated by $y_m = (Lg)(mQ)$.
(2) $\{q_i\}$ is optimized by minimizing $\sum_m |y_m + q_1 y_{m-1} + q_2 y_{m-2} + \cdots + q_H y_{m-H}|^2$.
(3) $\{z_i\}$ is calculated by solving $z^H + q_1 z^{H-1} + q_2 z^{H-2} + \cdots + q_H = 0$.
(4) $\{w_i\}$ is determined by $a(w_i) = (1/Q) \log z_i$.
(5) $\{c_i\}$ is optimized by minimizing $\sum_j (g(x_j) - \sum_i c_i \varphi_{w_i}(x_j))^2$.

Strictly speaking, $g(x)$ should be given for arbitrary $x$. However, in the practical application, if the number of training samples is sufficiently large so that $(Lg)(x)$ can be almost precisely approximated, this algorithm is available. In the third procedure, to solve the algebraic equation, the DKA method is applied, for example.

## 5  Experimental Results and Discussion

### 5.1  The backpropagation and the proposed method

For experiments, we used a probability density function and a regression function given by

$$Q(y|x) = \frac{1}{\sqrt{2\pi\sigma^2}} \exp(-\frac{(y - h(x))^2}{2\sigma^2})$$

$$h(x) = -\frac{1}{3} \tan^{-1}(\frac{x - 1/3}{0.04}) + \frac{1}{6} \tan^{-1}(\frac{x - 2/3}{0.02})$$

where $\sigma = 0.2$. One hundred input samples were set at the same interval in $[0,1)$, and output samples were taken from the above conditional distribution.

Table 1 shows the relation between the number of hidden units, training errors, and regression errors. In the table, the training error in the backpropagation shows the square error obtained after 100,000 training cycles. The training error in the proposed method shows the square error by the above algorithm. And the regression error shows the square error between the true regression curve $h(x)$ and the estimated curve.

Figure 2 shows the true and estimated regression lines: (0) the true regression line and sample points, (1) the estimated regression line with 2 hidden units, by the BP (the error backpropagation) after 100,000 training cycles, (2) the estimated regression line with 12 hidden units, by the BP after 100,000 training cycles, (3) the

Table 1: Training errors and regression errors

| Hidden | Backpropagation | | Proposed Method | |
|---|---|---|---|---|
| Units | Training | Regression | Training | Regression |
| 2 | 4.1652 | 0.7698 | 4.0889 | 0.3301 |
| 4 | 3.3464 | 0.4152 | 3.8755 | 0.2653 |
| 6 | 3.3343 | 0.4227 | 3.5368 | 0.3730 |
| 8 | 3.3267 | 0.4189 | 3.2237 | 0.4297 |
| 10 | 3.3284 | 0.4260 | 3.2547 | 0.4413 |
| 12 | 3.3170 | 0.4312 | 3.1988 | 0.5810 |

estimated line with 2 hidden units by the proposed method, and (4) the estimated line with 12 hidden units by the proposed method.

## 5.2 Discussion

When the number of hidden units was small, the training errors by the BP were smaller, but the regression errors were larger. When the number of hidden units was taken to be larger, the training error by the BP didn't decrease so much as the proposed method, and the regression error didn't increase so much as the proposed method.

By the error backpropagation, parameters didn't reach the maximum likelihood estimator, or they fell into local minima. However, when the number of hidden units was large, the neural network without the maximum likelihood estimator attained the better generalization. It seems that parameters in the local minima were closer to the true parameter than the maximum likelihood estimator.

Theoretically, in the case of the layered neural networks, the maximum likelihood estimator may not be subject to asymptotically normal distribution because the Fisher information matrix may be degenerate. This can be one reason why the experimental results contradict the ordinary statistical theory. Adding such a problem, the above experimental results show that the local minimum causes a strange problem. In order to construct the more precise learning theory for the backpropagation neural network, and to choose the better parameter for generalization, we maybe need a method to analyze learning and inference with a local minimum.

## 6    Conclusion

We have proposed solvable models of artificial neural networks, and studied their learning structure. It has been shown by the experimental results that the proposed method is useful in analysis of the neural network generalization problem.

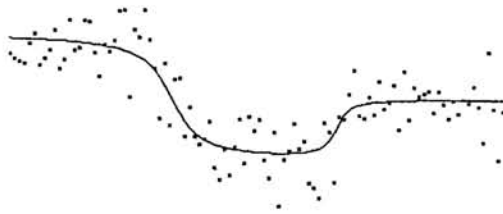

(0) True Curve and Samples.
Sample error sum = 3.6874

$H$ : the number of hidden units
$E_{train}$ : the training error
$E_{reg}$ : the regression error

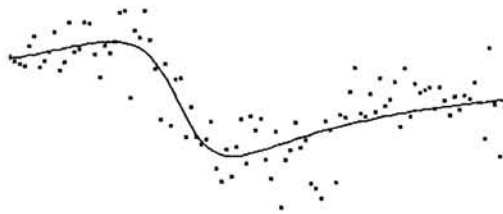

(1) BP after 100,000 cycles
$H = 2$, $E_{train} = 4.1652$, $E_{reg} = 0.7698$

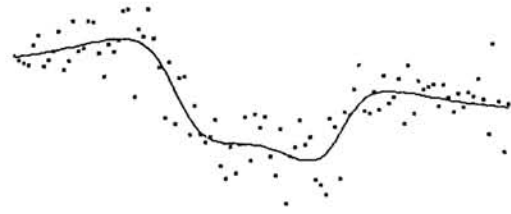

(2) BP after 100,000 cycles
$H = 12$, $E_{train} = 3.3170$, $E_{reg} = 0.4312$

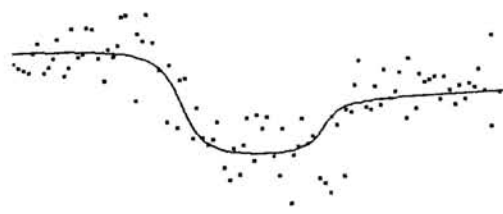

(3) Proposed Method
$H = 2$, $E_{train} = 4.0889$, $E_{reg} = 0.3301$

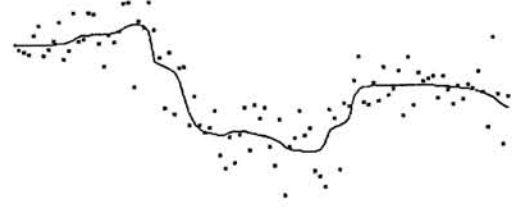

(4) Proposed Method
$H = 12$, $E_{train} = 3.1988$, $E_{reg} = 0.5810$

Figure 2: Experimental Results

## References

[1] H. White. (1989) Learning in artificial neural networks: a statistical perspective. *Neural Computation*, **1**, 425-464.

[2] N.Murata, S.Yoshizawa, and S.-I.Amari.(1992) Learning Curves, Model Selection and Complexity of Neural Networks. *Advances in Neural Information Processing Systems 5*, San Mateo, Morgan Kaufman, pp.607-614.

[3] R. J. Baxter. (1982) *Exactly Solved Models in Statistical Mechanics*, Academic Press.

[4] E. A. Coddington. (1955) *Theory of ordinary differential equations*, the McGraw-Hill Book Company, New York.

[5] S. Watanabe. (1993) Function approximation by neural networks and solution spaces of differential equations. Submitted to *Neural Networks*.